# History distribution matching method for predicting effectiveness of HIV combination therapies

**Jasmina Bogojeska**
Max-Planck Institute for Computer Science
Campus E1 4
66123 Saarbrücken, Germany
jasmina@mpi-inf.mpg.de

## Abstract

This paper presents an approach that predicts the effectiveness of HIV combination therapies by simultaneously addressing several problems affecting the available HIV clinical data sets: the different treatment backgrounds of the samples, the uneven representation of the levels of therapy experience, the missing treatment history information, the uneven therapy representation and the unbalanced therapy outcome representation. The computational validation on clinical data shows that, compared to the most commonly used approach that does not account for the issues mentioned above, our model has significantly higher predictive power. This is especially true for samples stemming from patients with longer treatment history and samples associated with rare therapies. Furthermore, our approach is at least as powerful for the remaining samples.

## 1    Introduction

According to [18], more than 33 million people worldwide are infected with the human immunodeficiency virus (HIV), for which there exists no cure. HIV patients are treated by administration of combinations of antiretroviral drugs, which succeed in suppressing the virus much longer than the monotherapies based on a single drug. Eventually, the drug combinations also become ineffective and need to be replaced. On such occasion, the very large number of potential therapy combinations makes the manual search for an effective therapy increasingly impractical. The search is particulary challenging for patients in the mid to late stages of antiretroviral therapy because of the accumulated drug resistance from all previous therapies. The availability of large clinical data sets enables the development of statistical methods that offer an automated procedure for predicting the outcome of potential antiretroviral therapies. An estimate of the therapy outcome can assist physicians in choosing a successful regimen for an HIV patient.

However, the HIV clinical data sets suffer from several problems. First of all, the clinical data comprise therapy samples that originate from patients with different treatment backgrounds. Also the various levels of therapy experience ranging from therapy-naïve to heavily pretreated are represented with different sample abundances. Second, the samples on different combination therapies have widely differing frequencies. In particular, many therapies are only represented with very few data points. Third, the clinical data do not necessarily have the complete information on all administered HIV therapies for all patients and the information on whether all administered therapies is available or not is also missing for many of the patients. Finally, the imbalance between the effective and the ineffective therapies is increasing over time: due to the knowledge acquired from HIV research and clinical practice the quality of treating HIV patients has largely increased in the recent years rendering the amount of effective therapies in recently collected data samples much larger than the amount of ineffective ones. These four problems create bias in the data sets which might negatively affect the usefulness of the derived statistical models.

In this paper we present an approach that addresses all these problems simultaneously. To tackle the issues of the uneven therapy representation and the different treatment backgrounds of the samples, we use information on both the current therapy and the patient's treatment history. Additionally, our method uses a distribution matching approach to account for the problems of missing information in the treatment history and the growing gap between the abundances of effective and ineffective HIV therapies over time. The performance of our *history distribution matching* approach is assessed by comparing it with two common reference methods in the so called *time-oriented validation scenario*, where all models are trained on data from the more distant past, while their performance is assessed on data from the more recent past. In this way we account for the evolving trends in composing drug combination therapies for treating HIV patients.

**Related work.** Various statistical learning methods, including artificial neural networks, decision trees, random forests, support vector machines (SVMs) and logistic regression [19, 11, 14, 10, 16, 1, 15], have been used to predict the effectiveness of HIV combination therapies from clinical data. None of these methods considers the problems affecting the available clinical data sets: different treatment backgrounds of the samples, uneven representations of therapies and therapy outcomes, and incomplete treatment history information. Some approaches [2, 4] deal with the uneven therapy representation by training a separate model for each combination therapy on all available samples with properly derived sample weights. The weights reflect the similarities between the target therapy and all training therapies. However, the therapy-specific approaches do not address the bias originating from the different treatment backgrounds of the samples, or the missing treatment history information.

## 2  Problem setting

Let $z$ denote a therapy sample that comprises the viral genotype $\mathbf{g}$ represented as a binary vector indicating the occurrence of a set of resistance-relevant mutations, the therapy combination $\mathbf{z}$ encoded as a binary vector that indicates the individual drugs comprising the current therapy, the binary vector $\mathbf{h}$ representing the drugs administered in all known previous therapies, and the label $y$ indicating the success (1) or failure ($-1$) of the therapy $\mathbf{z}$. Let $D = \{(\mathbf{g}_1, \mathbf{z}_1, \mathbf{h}_1, y_1), \ldots, (\mathbf{g}_m, \mathbf{z}_m, \mathbf{h}_m, y_m)\}$ denote the training set and let $s$ refer to the therapy sample of interest. Let $start(s)$ refer to the point of time when the therapy $\mathbf{s}$ was started and $patient(s)$ refer to the patient identifier corresponding to the therapy sample $s$. Then:

$$r(s) = \{\mathbf{z} \mid (start(z) \le start(s)) \text{ and } (patient(z) = patient(s))\}$$

denotes the complete treatment data associated with the therapy sample $s$ and will be referred to as *therapy sequence*. It contains all known therapies administered to $patient(s)$ not later than $start(s)$ ordered by their corresponding starting times. We point out that each therapy sequence also contains the current therapy, *i.e.,* the most recent therapy in the therapy sequence $r(s)$ is $\mathbf{s}$. Our goal is to train a model $f(\mathbf{g}, \mathbf{s}, \mathbf{h})$ that addresses the different types of bias associated with the available clinical data sets when predicting the outcome of the therapy $\mathbf{s}$. In the rest of the paper we denote the set of input features $(\mathbf{g}, \mathbf{s}, \mathbf{h})$ by $\mathbf{x}$.

## 3  History distribution matching method

The main idea behind the *history distribution matching method* we present in this paper is that the predictions for a given patient should originate from a model trained using samples from patients with treatment backgrounds similar as the one of the target patient. The details of this method are summarized in Algorithm 1. In what follows, we explain each step of this algorithm.

### 3.1  Clustering based on similarities of therapy sequences

Clustering partitions a set of objects into clusters, such that the objects within each cluster are more similar to one another than to the objects assigned to a different cluster [7]. In the first step of Algorithm 1, all available training samples are clustered based on the pairwise dissimilarity of their corresponding therapy sequences. In the following, we first describe a similarity measure for therapy sequences and then present the details of the clustering.

---
**Algorithm 1:** History distribution matching method

1. Cluster the training samples by using the pairwise dissimilarities of their corresponding therapy sequences.

2. For each (target) cluster:
   - Compute sample weights that match the distribution of all available training samples to the distribution of samples in the target cluster.
   - Train a sample-weighted logistic regression model using the sample weights computed in the previous distribution matching step.

---

**Similarity of therapy sequences.** In order to quantify the pairwise similarity of therapy sequences we use a slightly modified version of the *alignment similarity measure* introduced in [5]. It adapts sequence alignment techniques [13] to the problem of aligning therapy sequences by considering the specific therapies given to a patient, their respective resistance-relevant mutations, the order in which they were applied and the length of the therapy history. The alphabet used for the therapy sequence alignment comprises all distinct drug combinations making up the clinical data set. The pairwise similarities between the different drug combinations are quantified with the *resistance mutations kernel* [5], which uses the table of resistance-associated mutations of each drug afforded by the International AIDS society [8]. First, binary vectors indicating resistance-relevant mutations for the set of drugs occurring in a combination are calculated for each therapy. Then, the similarity score of two therapies of interest is computed as normalized inner product between their corresponding resistance mutation vectors. In this way, the therapy similarity also accounts for the similarity of the genetic fingerprint of the potential latent virus populations of the compared therapies. Each therapy sequence ends with the current (most recent) therapy – the one that determines the label of the sample and the sequence alignment is adapted such that the most recent therapies are always matched. Therefore, it also accounts for the problem of uneven representation of the different therapies in the clinical data. It has one parameter that specifies the linear gap cost penalty.

For the history distribution matching method, we modified the alignment similarity kernel described in the paragraph above such that it also takes the importance of the different resistance-relevant mutations into account. This is achieved by updating the resistance mutations kernel, where instead of using binary vectors that indicate the occurrence of a set of resistance-relevant mutations, we use vectors that indicate their importance. If two or more drugs from a certain drug group, that comprise a target therapy share a resistance mutation, then we consider its maximum importance score. Importance scores for the resistance-relevant mutations are derived from in-vivo experiments and can be obtained from the Stanford University HIV Drug Resistance Database [12]. Furthermore, we want to keep the cluster similarity measure parameter-free, such that in the process of model selection the clustering Step 1 in Algorithm 1 is decoupled from the Step 2 and is computed only once. This is achieved by computing the alignments with zero gap costs and ensures time-efficient model selection procedure. However, in this case only the similarities of the matched therapies comprising the two compared therapy sequences contribute to the similarity score and thus the differing lengths of the therapy sequences are not accounted for. Having a clustering similarity measure that addresses the differing therapy lengths is important for tackling the uneven sample representation with respect to the level of therapy experience. In order to achieve this we normalize each pairwise similarity score with the length of the longer therapy sequence. This yields pairwise similarity values in the interval $[0, 1]$ which can easily be converted to dissimilarity values in the same range by subtracting them from 1.

**Clustering.** Once we have a measure of dissimilarity of therapy sequences, we cluster our data using the most popular version of $K$-medoids clustering [7], referred to as *partitioning around medoids* (PAM) [9]. The main reason why we choose this approach instead of the simpler $K$-means clustering [7] is that it can use any precomputed dissimilarity matrix. We select the number of clusters with the *silhouette validation technique* [17], which uses the so-called *silhouette value* to assess the quality of the clustering and select the optimal number of clusters.

## 3.2 Cluster distribution matching

The clustering step of our method groups the training data into different bins based on their therapy sequences. However, the complete treatment history is not necessarily available for all patients in our clinical data set. Therefore, by restricting the prediction model for a target sample only to the data from its corresponding cluster, the model might ignore relevant information from the other clusters. The approach we use to deal with this issue is inspired by the multi-task learning with distribution matching method introduced in [2].

In our current problem setting, the goal is to train a prediction model $f_c : \mathbf{x} \to y$ for each cluster $c$ of similar treatment sequences, where $\mathbf{x}$ denotes the input features and $y$ denotes the label. The straightforward approach to achieve this is to train a prediction model by using only the samples in cluster $c$. However, since the available treatment history for some samples might be incomplete, totally excluding the samples from all other clusters ($\neq c$) ignores relevant information about the model $f_c$. Furthermore, the cluster-specific tasks are related and the samples from the other clusters – especially those close to the cluster boundaries of cluster $c$ – also carry valuable information for the model $f_c$. Therefore, we use a multi-task learning approach where a separate model is trained for each cluster by not only using the training samples from the target cluster, but also the available training samples from the remaining clusters with appropriate sample-specific weights. These weights are computed by matching the distribution of all samples to the distribution of the samples of the target cluster and they thereby reflect the relevance of each sample for the target cluster. In this way, the model for the target cluster uses information from the input features to extract relevant knowledge from the other clusters.

More formally, let $D = \{(\mathbf{x}_1, y_1, c_1), \dots, (\mathbf{x}_m, y_m, c_m)\}$ denote the training data, where $c_i$ denotes the cluster associated with the training sample $(\mathbf{x}_i, y_i)$ in the history-based clustering. The training data are governed by the joint training distribution $\sum_c p(c)p(\mathbf{x}, y|c)$. The most accurate model for a given target cluster $t$ minimizes the loss with respect to the conditional probability $p(\mathbf{x}, y|t)$ referred to as the *target distribution*. In [2] it is shown that:

$$E_{(\mathbf{x},y) \sim p(\mathbf{x},y|t)}[\ell(f_t(\mathbf{x}))] = E_{(\mathbf{x},y) \sim \sum_c p(c)p(\mathbf{x},y|c)}[r_t(\mathbf{x}, y)\ell(f_t(\mathbf{x}))], \tag{1}$$

where:

$$r_t(\mathbf{x}, y) = \frac{p(\mathbf{x}, y|t)}{\sum_c p(c)p(\mathbf{x}, y|c)}. \tag{2}$$

In other words, by using sample-specific weights $r_t(\mathbf{x}, y)$ that match the training distribution $\sum_c p(c)p(\mathbf{x}, y|c)$ to the target distribution $p(\mathbf{x}, y|t)$ we can minimize the expected loss with respect to the target distribution by minimizing the expected loss with respect to the training distribution. The weighted training data are governed by the correct target distribution $p(\mathbf{x}, y|t)$ and the sample weights reflect the relevance of each training sample for the target model. The weights are derived based on information from the input features. If a sample was assigned to the wrong cluster due to the incompleteness of the treatment history, by matching the training to the target distribution it can still receive high sample weight for the model of its correct cluster.

In order to avoid the estimation of the high-dimensional densities $p(\mathbf{x}, y|t)$ and $p(\mathbf{x}, y|c)$ in Equation 2, we follow the example of [3, 2] and compute the sample weights $r_t(\mathbf{x}, y)$ using a discriminative model for a conditional distribution with a single variable:

$$r_t(\mathbf{x}, y) = \frac{p(t|\mathbf{x}, y)}{p(t)}, \tag{3}$$

where $p(t|\mathbf{x}, y)$ quantifies the probability that a sample $(\mathbf{x}, y)$ randomly drawn from the training set $D$ belongs to the target cluster $t$. $p(t)$ is the prior probability which can easily be estimated from the training data.

As in [2], $p(t|\mathbf{x}, y)$ is modeled for all clusters jointly using a kernelized version of multi-class logistic regression with a feature mapping that separates the effective from the ineffective therapies:

$$\Phi(\mathbf{x}, y) = \begin{bmatrix} \delta(y, +1)\mathbf{x} \\ \delta(y, -1)\mathbf{x} \end{bmatrix}, \tag{4}$$

where $\delta$ is the Kronecker delta ($\delta(a, b) = 1$, if $a = b$, and $\delta(a, b) = 0$, if $a \neq b$). In this way, we can train the cluster-discriminative models for the effective and the ineffective therapies independently,

and thus, by proper time-oriented model selection address the increasing imbalance in their representation over time. Formally, the multi-class model is trained by maximizing the log-likelihood over the training data using a Gaussian prior on the model parameters:

$$\arg \max_{\mathbf{v}} \sum_{(\mathbf{x}_i, y_i, c_i) \in D_c} \log(p(c_i | \mathbf{x}_i, y_i, \mathbf{v})) + \mathbf{v}^\mathsf{T} \Sigma^{-1} \mathbf{v},$$

where $\mathbf{v}$ are the model parameters (a concatenation of the cluster specific parameters $\mathbf{v}_c$), and $\Sigma$ is the covariance matrix of the Gaussian prior.

### 3.3  Sample-weighted logistic regression method

As described in the previous subsection, we use a multi-task distribution matching procedure to obtain sample-specific weights for each cluster, which reflect the relevance of each sample for the corresponding cluster. Then, a separate logistic regression model that uses all available training data with the proper sample weights is trained for each cluster. More formally, let $t$ denote the target cluster and let $r_t(\mathbf{x}, y)$ denote the weight of the sample $(\mathbf{x}, y)$ for the cluster $t$. Then, the prediction model for the cluster $t$ that minimizes the loss over the weighted training samples is given by:

$$\arg \min_{\mathbf{w}_t} \frac{1}{|D|} \sum_{(\mathbf{x}_i, y_i) \in D} r_t(x_i, y)^\gamma \cdot \ell(f_t(\mathbf{x}_i), y_i) + \sigma \mathbf{w}_t^T \mathbf{w}_t, \tag{5}$$

where $\mathbf{w}_t$ are the model parameters, $\sigma$ is the regularization parameter, $\gamma$ is a smoothing parameter for the sample-specific weights and $\ell(f(\mathbf{x}, \mathbf{w}_t), y) = \ln(1 + \exp(-y\mathbf{w}_t^T \mathbf{x}))$ is the loss of linear logistic regression.

All in all, our method first clusters the training data based on their corresponding therapy sequences and then learns a separate model for each cluster by using relevant data from the remaining clusters. By doing so it tackles the problems of the different treatment backgrounds of the samples and the uneven sample representation in the clinical data sets with respect to the level of therapy experience. Since the alignment kernel considers the most recent therapy and the drugs comprising this therapy are encoded as a part of the input feature space, our method also deals with the differing therapy abundances in the clinical data sets. Once we have the models for each cluster, we use them to predict the label of a given test sample $\mathbf{x}$ as follows: First of all, we use the therapy sequence of the target sample to calculate its dissimilarity to the therapy sequences of each of the cluster centers. Then, we assign the sample $\mathbf{x}$ to the cluster $c$ with the closest cluster center. Finally, we use the logistic regression model trained for cluster $c$ to predict the label $y$ for the target sample $\mathbf{x}$.

## 4  Experiments and results

### 4.1  Data

The clinical data for our model are extracted from the EuResist [16] database that contains information on 93014 antiretroviral therapies administered to 18325 HIV (subtype B) patients from several countries in the period from 1988 to 2008. The information employed by our model is extracted from these data: the viral sequence $\mathbf{g}$ assigned to each therapy sample is obtained shortly before the respective therapy was started (up to 90 days before); the individual drugs of the currently administered therapy $\mathbf{z}$; all available (known) therapies administered to each patient $\mathbf{h}$, $r(z)$; and the response to a given therapy quantified with a label $y$ (success or failure) based on the virus load values (copies of viral RNA per $ml$ blood plasma) measured during its course (for more details see [4] and the Supplementary material). Finally, our training set comprises 6537 labeled therapy samples from 690 distinct therapy combinations.

### 4.2  Validation setting

**Time-oriented validation scenario.**  The trends of treating HIV patients change over time as a result of the gathered practical experience with the drugs and the introduction of new antiretroviral drugs. In order to account for this phenomenon we use the *time-oriented validation scenario* [4] which makes a time-oriented split when selecting the training and the test set. First, we order all

available training samples by their corresponding therapy starting dates. We then make a time-oriented split by selecting the most recent 20% of the samples as the test set and the rest as the training set. For the model selection we split the training set further in a similar manner. We take the most recent 25% of the training set for selecting the best model parameters (see Supplementary material) and refer to this set as tuning set. In this way, our models are trained on the data from the more distant past, while their performance is measured on the data from the more recent past. This scenario is more realistic than other scenarios since it captures how a given model would perform on the recent trends of combining the drugs. The details of the data sets resulting from this scenario are given in Table 1, where one can also observe the large gap between the abundances of the effective and ineffective therapies, especially for the most recent data.

Table 1: Details on the data sets generated in the time-oriented validation scenario.

| Data set | training | tuning | test |
|---|---|---|---|
| Sample count | 3596 | 1634 | 1307 |
| Success rate | 69% | 79% | 83% |

The search for an effective HIV therapy is particulary challenging for patients in the mid to late stages of antiretroviral therapy when the number of therapy options is reduced and effective therapies are increasingly hard to find because of the accumulated drug resistance mutations from all previous therapies. The therapy samples gathered in the HIV clinical data sets are associated with patients whose treatment histories differ in length: while some patients receive their first antiretroviral treatment, others are heavily pretreated. These different sample groups, from treatment naïve to heavily pretreated, are represented unevenly in the HIV clinical data with fewer samples associated to therapy-experienced patients (see Figure 1 (a) in the Supplementary material). In order to assess the ability of a given target model to address this problem, we group the therapy samples in the test set into different bins based on the number of therapies administered prior to the therapy of interest – the current therapy (see Table 1 in the Supplementary material). Then, we assess the quality of a given target model by reporting its performance for each of the bins. In this way we can assess the predictive power of the models in dependence on the level of therapy experience.

Another important property of an HIV model is its ability to address the uneven representation of the different therapies (see Figure 1 (b) in the Supplementary material). In order to achieve this we group the therapies in the test set based on the number of samples they have in the training set, and then we measure the model performance on each of the groups. The details on the sample counts in each of the bins are given in Table 2 of the Supplementary material. In this manner we can evaluate the performance of the models for the rare therapies. Due to the lack of data and practical experience for the rare HIV combination therapies, predicting their efficiency is more challenging compared to estimating the efficiency of the frequent therapies.

**Reference methods.** In our computational experiments we compare the results of our history distribution matching approach, denoted as *transfer history clustering validation scenario*, to those of three reference approaches, namely the *one-for-all validation scenario*, the *history-clustering validation scenario*, and the *therapy-specific validation scenario*. The one-for-all method mimics the most common approaches in the field [16, 1, 19] that train a single model (here logistic regression) on all available therapy samples in the data set. The information on the individual drugs comprising the target (most recent) therapy and the drugs administered in all its available preceding therapies are encoded in a binary vector and supplied as input features. The history-clustering method implements a modified version of Algorithm 1 that skips the distribution matching step. In other words, a separate model is trained for each cluster by using only the data from the respective cluster. We introduce this approach to assess the importance of the distribution matching step. The therapy-specific scenario implements the drugs kernel therapy similarity model described in [4]. It represents the approaches that train a separate model for each combination therapy by using not only the samples from the target therapy but also the available samples from similar therapies with appropriate sample-importance weights.

**Performance measures.** The performance of all considered methods is assessed by reporting their corresponding accuracies (ACC) and AUCs (Area Under the ROC Curve). The accuracy reflects the ability of the methods to make correct predictions, *i.e.,* to discriminate between successful and failing HIV combination therapies. With the AUC we are able to assess the quality of the ranking based

on the probability of therapy success. For this reason, we carry out the model selection based on both accuracy and AUC and then use accuracy or AUC, respectively, to assess the model performance. In order to compare the performance of two methods on a separate test set, the significance of the difference of two accuracies as well as their standard deviations are calculated based on a paired t-test. The standard deviations of the AUC values and the significance of the difference of two AUCs used for the pairwise method comparison are estimated as described in [6].

## 4.3 Experimental results

According to the results from the silhouette validation technique [17] displayed in Figure 2 in the Supplementary material, the first clustering step of Algorithm 1 divides our training data into two clusters – one comprises the samples with longer therapy sequences (with average treatment history length of $5.507$ therapies), and the other one those with shorter therapy sequences (with average treatment history length of $0.308$ therapies). Thus, the transfer history distribution matching method trains two models, one for each cluster. The clustering results are depicted in Figure 3 in the Supplementary material. In what follows, we first present the results of the time-oriented validation scenario stratified for the length of treatment history, followed by the results stratified for the abundance of the different therapies. In both cases we report both the accuracies and the AUCs for all considered methods.

The computational results for the transfer history method and the three reference methods stratified for the length of the therapy history are summarized in Figure 1, where (a) depicts the accuracies, and (b) depicts the AUCs. For samples with a small number ($\leq 5$) of previously administered therapies, *i.e.,* with short treatment histories, all considered models have comparable accuracies. For test samples from patients with longer ($> 5$) treatment histories, the transfer history clustering approach achieves significantly better accuracy (*p-values* $\leq 0.004$) compared to those of the reference methods. According to the paired difference test described in [6], the transfer history approach has significantly better AUC performance for test samples with longer ($> 5$) treatment histories compared to the one-for-all (*p-value* $= 0.043$) and the history-clustering (*p-value* $= 0.044$) reference methods. It also has better AUC performance compared to the one of the therapy-specific model, yet this improvement is not significant (*p-value* $= 0.253$). Furthermore, the transfer history approach achieves better AUCs for test samples with less than five previously administered therapies compared to all reference methods. However, the improvement is only significant for the one-for-all method (*p-value* $= 0.007$). The corresponding *p-values* for the history-clustering method and the therapy-specific method are $0.080$ and $0.178$, respectively.

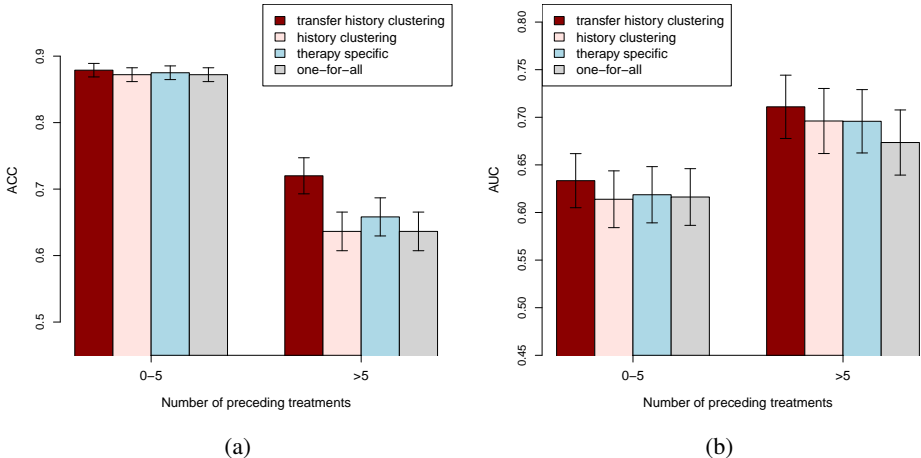

(a)  (b)

Figure 1: Accuracy (a) and AUC (b) results of the different models obtained on the test set in the time-oriented validation scenario. Error bars indicate the standard deviations of each model. The test samples are grouped based on their corresponding number of known previous therapies.

The experimental results, stratified for the abundance of the therapies summarizing the accuracies and AUCs for all considered methods, are depicted in Figure 2 (a) and (b), respectively. As can

be observed from Figure 2 (a), all considered methods have comparable accuracies for the test therapies with more than seven samples. The transfer history method achieves significantly better accuracy (*p-values* $\leq$ 0.0001) compared to all reference methods for the test therapies with few $(0 - 7)$ available training samples. Considering the AUC results in Figure 2 (b), the transfer history approach outperforms all the reference models for the rare test therapies (with $0 - 7$ training samples) with estimated *p-values* of 0.05 for the one-for-all, 0.042 for the therapy-specific and 0.1 for the history-clustering model. The one-for-all and the therapy-specific models have slightly better AUC performance compared to the transfer history and the history-clustering approaches for test therapies with $8 - 30$ available training samples. However, according to the paired difference test described in [6], the improvements are not significant with *p-values* larger than 0.141 for all pairwise comparisons. Moreover, considering the test therapies with more than 30 training samples the transfer history approach significantly outperforms the one-for-all approach with estimated *p-value* of 0.037. It also has slightly better AUC performance than the history-clustering model and the therapy-specific model, however these improvements are not significant with estimated *p-values* of 0.064 and 0.136, respectively.

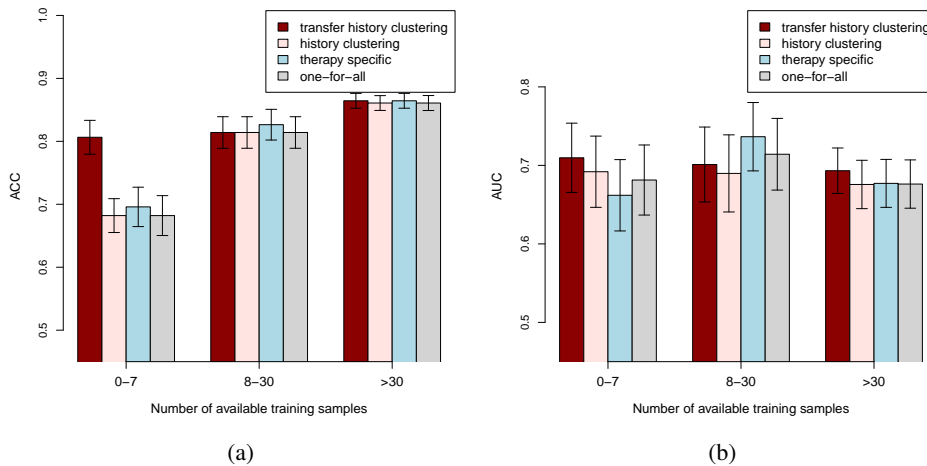

Figure 2: Accuracy (a) and AUC (b) results of the different models obtained on the test set in the time-oriented validation scenario. Error bars indicate the standard deviations of each model. The test samples are grouped based on the number of available training examples for their corresponding therapy combinations.

## 5   Conclusion

This paper presents an approach that simultaneously considers several problems affecting the available HIV clinical data sets: the different treatment backgrounds of the samples, the uneven representation of the different levels of therapy experience, the missing treatment history information, the uneven therapy representation and the unbalanced therapy outcome representation especially pronounced in recently collected samples. The transfer history clustering model has its prime advantage for samples stemming from patients with long treatment histories and for samples associated with rare therapies. In particular, for these two groups of test samples it achieves significantly better accuracy than all considered reference approaches. Moreover, the AUC performance of our method for these test samples is also better than all reference methods and significantly better compared to the one-for-all method. For the remaining test samples both the accuracy and the AUC performance of the transfer history method are at least as good as the corresponding performances of all considered reference methods.

**Acknowledgments**

We gratefully acknowledge the EuResist EEIG for providing the clinical data. We thank Thomas Lengauer for the helpful comments and for supporting this work. We also thank Levi Valgaerts for the constructive suggestions. This work was funded by the Cluster of Excellence (Multimodal Computing and Interaction).

# References

[1] A. Altmann, M. Däumer, N. Beerenwinkel, E. Peres, Y. Schülter, A. Büch, S. Rhee, A. Sönnerborg, WJ. Fessel, M. Shafer, WR. Zazzi, R. Kaiser, and T. Lengauer. Predicting response to combination antiretroviral therapy: retrospective validation of geno2pheno-THEO on a large clinical database. *Journal of Infectious Diseases*, 199:999–1006, 2009.

[2] S. Bickel, J. Bogojeska, T. Lengauer, and T. Scheffer. Multi-task learning for HIV therapy screening. In *Proceedings of the International Conference on Machine Learning*, 2008.

[3] S. Bickel, M. Brückner, and T. Scheffer. Discriminative learning for differing training and test distributions. In *Proceedings of the International Conference on Machine Learning*, 2007.

[4] J. Bogojeska, S. Bickel, A. Altmann, and T. Lengauer. Dealing with sparse data in predicting outcomes of HIV combination therapies. *Bioinformatics*, 26:2085–2092, 2010.

[5] J. Bogojeska, D. Stöckel, M. Zazzi, R. Kaiser, F. Incardona, M. Rosen-Zvi, and T. Lengauer. History-alignment models for bias-aware prediction of virological response to HIV combination therapy. *submitted*, 2011.

[6] J. Hanley and B. McNeil. A method of comparing the areas under receiver operating characteristic curves derived from the same cases. *Radiology*, 148:839–843, 1983.

[7] T. Hastie, R. Tibshirani, and J. Friedman. *The Elements of Statistical Learning*. Springer, 2009.

[8] VA. Johnson, F. Brun-Vezinet, B. Clotet, HF. Günthrad, DR. Kuritzkes, D. Pillay, JM. Schapiro, and DD. Richman. Update of the drug resistance mutations in HIV-1: December 2008. *Topics in HIV Medicine*, 16:138–145, 2008.

[9] L. Kaufman and PJ. Rousseeuw. *Finding Groups in Data. An introduction to cluster analysis*. John Wiley and Sons, Inc., 1990.

[10] B. Larder, D. Wang, A. Revell, J. Montaner, R. Harrigan, F. De Wolf, J. Lange, S. Wegner, L. Ruiz, MJ. Prez-Elas, S. Emery, J. Gatell, A. DArminio Monforte, C. Torti, M. Zazzi, and C. Lane. The development of artificial neural networks to predict virological response to combination HIV therapy. *Antiviral Therapy*, 12:15–24, 2007.

[11] RH. Lathrop and MJ. Pazzani. Combinatorial optimization in rapidly mutating drug-resistant viruses. *Journal of Combinatorial Optimization*, 3:301–320, 1999.

[12] TF. Liu and Shafer RW. Web resources for HIV type 1 genotypic-resistance test interpretation. *Clinical Infectious Diseases*, 42, 2006.

[13] S. Needleman and C. Wunsch. A general method applicable to the search for similarities in the amino acid sequence of two proteins. *Journal of Molecular Biology*, 48(3):443–453, 1970.

[14] DA. Ouattara. Mathematical analysis of the HIV-1 infection: parameter estimation, therapies effectiveness and therapeutical failures. In *Engineering in Medicine and Biology Society*, 2005.

[15] M. Prosperi, A. Altmann, M. Rosen-Zvi, E. Aharoni, G. Borgulya, F. Bazso, A. Sönnerborg, E. Schülter, D. Struck, G. Ulivi, A. Vandamme, J. Vercauteren, and M. Zazzi. Investigation of expert rule bases, logistic regression, and non-linear machine learning techniques for predicting response to antiretroviral treatment. *Antiviral Therapy*, 14:433–442, 2009.

[16] M. Rosen-Zvi, A. Altmann, M. Prosperi, E. Aharoni, H. Neuvirth, A. Sönnerborg, E. Schülter, D. Struck, Y. Peres, F. Incardona, R. Kaiser, M. Zazzi, and T. Lengauer. Selecting anti-HIV therapies based on a variety of genomic and clinical factors. *Proceedings of the ISMB*, 2008.

[17] P. J. Rousseeuw. Silhouettes: a graphical aid to the interpretation and validation of cluster analysis. *Journal of Computational and Applied Mathematics*, 20:53–65, 1987.

[18] UNAIDS/WHO. Report on the global aids epidemic: 2010. 2010.

[19] D. Wang, BA. Larder, A. Revell, R. Harrigan, and J. Montaner. A neural network model using clinical cohort data accurately predicts virological response and identifies regimens with increased probability of success in treatment failures. *Antiviral Therapy*, 8:U99–U99, 2003.

